# Correlated Neuronal Response: Time Scales and Mechanisms

**Wyeth Bair**
Howard Hughes Medical Inst.
NYU Center for Neural Science
4 Washington Pl., Room 809
New York, NY 10003

**Ehud Zohary**
Dept. of Neurobiology
Institute of Life Sciences
The Hebrew University, Givat Ram
Jerusalem, 91904 ISRAEL

**Christof Koch**
Computation and Neural Systems
Caltech, 139-74
Pasadena, CA 91125

## Abstract

We have analyzed the relationship between correlated spike count and the peak in the cross-correlation of spike trains for pairs of simultaneously recorded neurons from a previous study of area MT in the macaque monkey (Zohary et al., 1994). We conclude that common input, responsible for creating peaks on the order of ten milliseconds wide in the spike train cross-correlograms (CCGs), is also responsible for creating the correlation in spike count observed at the two second time scale of the trial. We argue that both common excitation and inhibition may play significant roles in establishing this correlation.

## 1 INTRODUCTION

In a previous study of pairs of MT neurons recorded using a single extracellular electrode, it was found that the spike count during two seconds of visual motion stimulation had an average correlation coefficient of $r = 0.12$ and that this correlation could significantly limit the usefulness of pooling across increasingly large populations of neurons (Zohary et al., 1994). However, correlated spike count between two neurons could in principle occur at several time-scales. Correlated drifts

in the excitability of the cells, for example due to normal biological changes or electrode induced changes, could cause correlation at a time scale of many minutes. Alternatively, attentional or priming effects from higher areas could change the responsivity of the cells at the time scale of an experimental trial. Or, as suggested here, common input that changes on the order of milliseconds could cause correlation in spike count. The first section determines the time scale at which the neurons are correlated by analyzing the relationship between the peak in the spike train cross-correlograms (CCGs) and the correlation between the spike counts using a construct we call the *trial* CCG. The second section examines temporal structure that is indicative of correlated suppression of firing, perhaps due to inhibition, which may also contribute to the spike count correlation.

## 2 THE TIME SCALE OF CORRELATION

At the time scale of the single trial, the correlation, $r_{sc}$, of spike counts $x$ and $y$ from two neurons recorded during nominally identical two second stimuli was computed using Pearson's correlation coefficient,

$$r_{sc} = \frac{\mathrm{E}[xy] - \mathrm{E}x\mathrm{E}y}{\sigma_x \sigma_y}, \tag{1}$$

where E is expected value and $\sigma^2$ is variance. If spike counts are converted to $z$-scores, i.e., zero mean and unity variance, then $r_{sc} = \mathrm{E}[xy]$, and $r_{sc}$ may be interpreted as the zero-lag value of the cross-correlation of the $z$-scored spike counts. The *trial* CCGs resulting from this procedure are shown for two pairs of neurons in Fig. 1.

To distinguish between cases like the two shown in Fig. 1, the correlation was broken into a long-term component, $r_{lt}$, the average value (computed using a Gaussian window of standard deviation 4 trials) surrounding the zero-lag value, and a short-term component, $r_{st}$, the difference between the zero-lag value and $r_{lt}$. Across 92 pairs of neurons from three monkeys, the average $r_{st}$ was 0.10 (s.d. 0.17) while $r_{lt}$ was not significantly different from zero (mean 0.01, s.d. 0.11). The mean of $r_{st}$ was similar to the overall correlation of 0.12 reported by Zohary et al. (1994).

Under certain assumptions, including that the time scale of correlation is less than the trial duration, $r_{st}$ can be estimated from the area under the spike train CCG and the areas under the autocorrelations (derivation omitted). Under the additional assumption that the spike trains are individually Poisson and have no peak in the autocorrelation except that which occurs by definition at lag zero, the correlation coefficient for spike count can be estimated by

$$r_{peak} \approx \sqrt{\lambda_A \lambda_B} \mathrm{Area}, \tag{2}$$

where $\lambda_A$ and $\lambda_B$ are the mean firing rates of neurons A and B, and *Area* is the area under the spike train CCG peak, like that shown in Fig. 2 for one pair of neurons. Taking *Area* to be the area under the CCG between $\pm 32$ msec gives a good estimate of short-term $r_{st}$, as shown in Fig. 3. In addition to the strong correlation ($r = 0.71$) between $r_{peak}$ and $r_{st}$, $r_{peak}$ is a less noisy measure, having standard deviation (not shown) on average one fourth as large as those of $r_{st}$.

We conclude that the common input that causes the peaks in the spike train CCGs is also responsible for the correlation in spike count that has been previously reported.

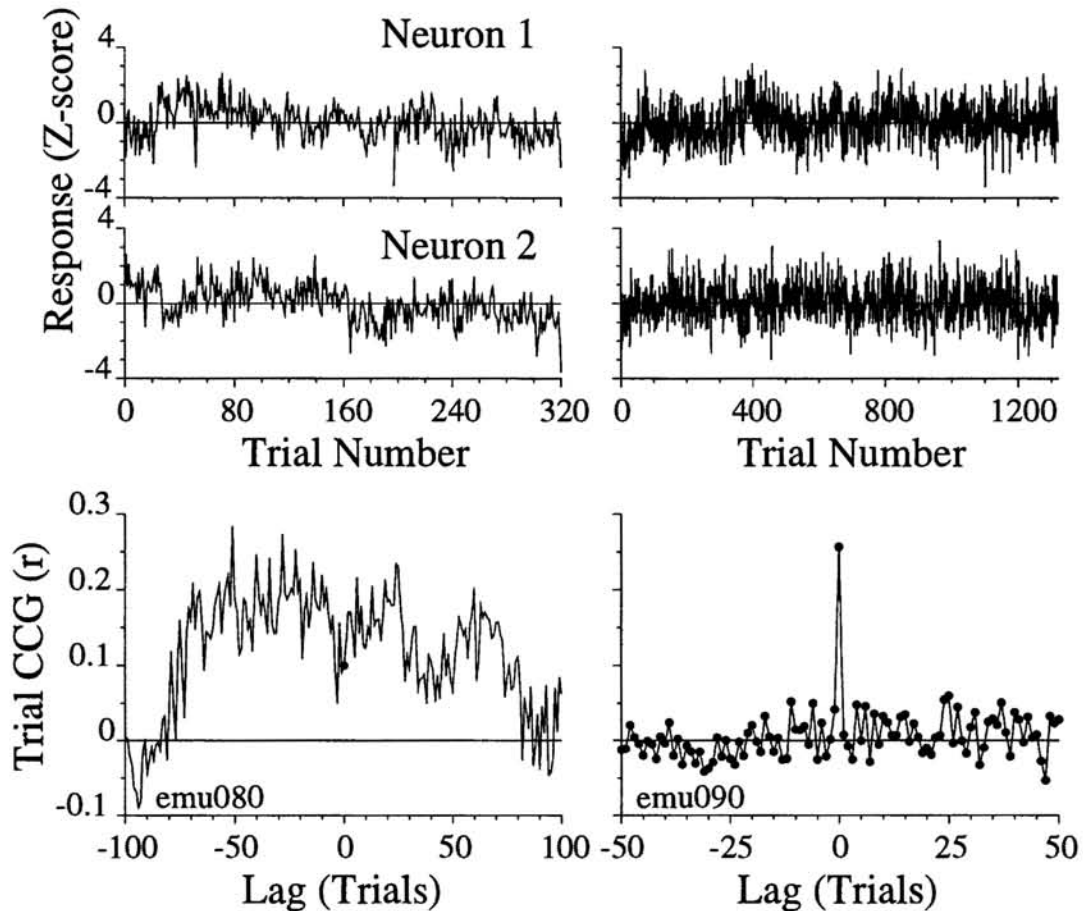

Figure 1: Normalized responses for two pairs of neurons and their trial cross-correlograms (CCGs). The upper traces show the $z$-scored spike counts for all trials in the order they occurred. Spikes were counted during the 2 sec stimulus, but trials occurred on average 5 sec apart, so 100 trials represents about 2.5 minutes. The lower traces show the trial CCGs. For the pair of cells in the left panel, responsivity drifts during the experiment. The CCG (lower left) shows that the drift is correlated between the two neurons over nearly 100 trials. For the pair of cells in the right panel, the trial CCG shows a strong correlation only for simultaneous trials. Thus, the measured correlation coefficient (trial CCG at zero lag) seems to occur at a long time scale on the left but a short time scale (less than or equal to one trial) on the right. The zero-lag value can be broken into two components, $r_{st}$ and $r_{lt}$ (short term and long term, respectively, see text). The short-term component, $r_{st}$, is the value at zero lag minus the weighted average value at surrounding lag times. On the left, $r_{st} \approx 0$, while on the right, $r_{lt} \approx 0$.

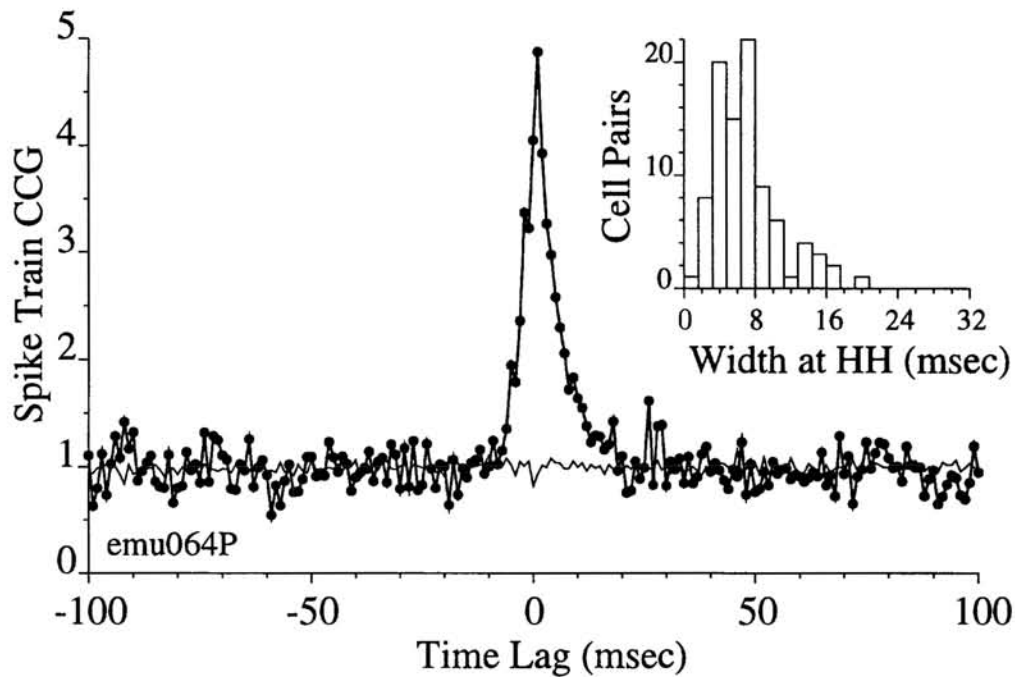

Figure 2: A spike train CCG with central peak. The frequency histogram of widths at half-height is shown (inset) for 92 cell pairs from three monkeys. The area of the central peak measured between ±32 msec is used to predict the correlation coefficients, $r_{peak}$, plotted in Fig. 3. The $y$-axis indicates the probability of a coincidence relative to that expected for Poisson processes at the measured firing rates.

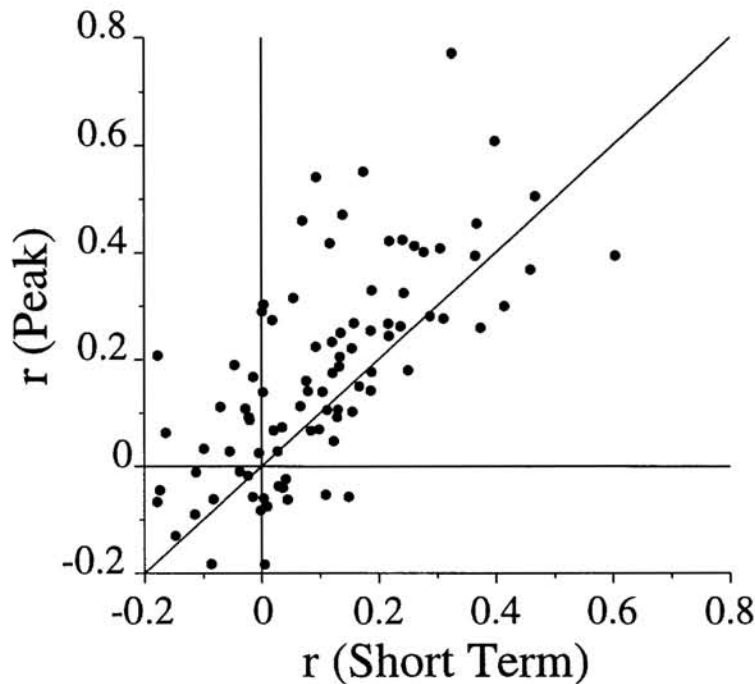

Figure 3: The area of the peak of the spike train CCG yields a prediction, $r_{peak}$ (see Eqn. 2), that is strongly correlated ($r = 0.71$, $p < 0.00001$), with the short-term spike count correlation coefficient, $r_{st}$. The absence of points in the lower right corner of the plot indicates that there are no cases of a pair of cells being strongly correlated without having a peak in the spike train CCG.

In Fig. 3, there are no pairs of neurons that have a short-term correlation and yet do not have a peak in the ±32 msec range of the spike train CCG.

## 3 CORRELATED SUPPRESSION

There is little doubt that common excitatory input causes peaks like the one shown in Fig. 2 and therefore results in the correlated spike count at the time scale of the trial. However, we have also observed correlated periods of suppressed firing that may point to inhibition as another contribution to the CCG peaks and consequently to the correlated spike count.

Fig. 4 **A** and **B** show the response of one neuron to coherent preferred and null direction motion, respectively. Excessively long inter-spike intervals (ISIs), or *gaps*, appear in the response to preferred motion, while bursts appear in the response to null motion. Across a database of 84 single neurons from a previous study (Britten et al., 1992), the occurrence of the gaps and bursts has a symmetrical time course—both are most prominent on average from 600–900 msec post-stimulus onset, although there are substantial variations from cell to cell (Bair, 1995). The gaps, roughly 100 msec long, are not consistent with the slow, steady adaptation (presumably due to potassium currents) which is observed under current injection in neocortical pyramidal neurons, e.g., the $RS_1$ and $RS_2$ neurons of Agmon and Connors (1992).

Fig. 4 **C** shows spike trains from two simultaneously recorded neurons stimulated with preferred direction motion. The longest gaps appear to occur at about the same time. To assess the correlation with a cross-correlogram, we first transform the spike trains to *interval* trains, shown in Fig. 4 **D** for the spike trains in **C**. This emphasizes the presence of long ISIs and removes some of the information regarding the precise occurrence times of action potentials. The interval cross-correlation (ICC) between each pair of interval trains is computed and averaged over all trials, and the average shift predictor is subtracted. Fig. 4 **E** and **F** show ICCs (thick lines) for two different pairs of neurons. In 17 of 31 pairs (55%), there were peaks in the raw ICC that were at least 4 standard errors above the level of the shift predictor. The peaks were on average centered (mean 4.3 msec, SD 54 msec) and had mean width at half-height of 139 msec (SD 59 msec).

To isolate the cause of the peaks, the *long* intervals in the trains were set to the mean of the *short* intervals. Long intervals were defined as those that accounted for 30% of the duration of the data and were longer than all short intervals. Note that this is only a small fraction of the number of ISIs in the spike train (typically less than about 10%), since a few long intervals consume the same amount of time as many short intervals. Data from 300–1950 msec was processed, avoiding the on-transient and the lack of final interval. With the longest intervals neutralized, the peaks were pushed down to the level of the noise in the ICC (thin lines, Fig. 4 **E**, **F**). Thus, 90% of the action potentials may serve to set a mean rate, while a few periods of long ISIs dominate the ICC peaks.

The correlated gaps are consistent with common inhibition to neurons in a local region of cortex, and this inhibition adds area to the spike train CCG peaks in the form of a broader base (not shown). The data analyzed here is from behaving animals, so the gaps may be related to small saccades (within the 0.5 degree

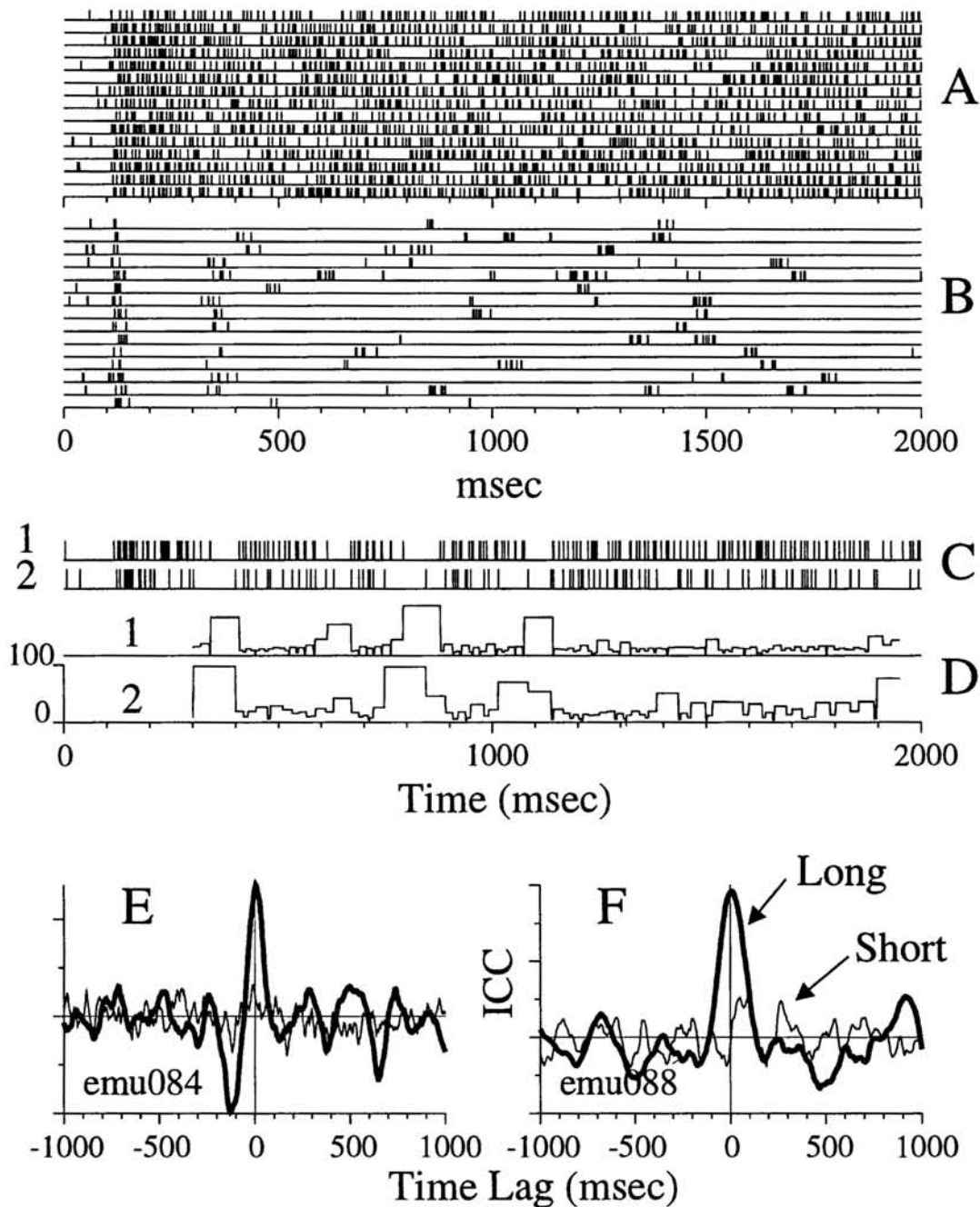

Figure 4: **(A)** The brisk response to coherent preferred direction motion is interrupted by occasional excessively long inter-spike intervals, i.e., gaps. **(B)** The suppressed response to null direction motion is interrupted by bursts of spikes. **(C)** Simultaneous spike trains from two neurons show correlated gaps in the preferred direction response. **(D)** The interval representation for the spike trains in **C**. **(E,F)** Interval cross-correlograms have peaks indicating that the gaps are correlated (see text).

fixation window) or eyelid blink. It has been hypothesized that blink suppression and saccadic visual suppression may operate through the same pathways and are of neuronal origin (Ridder and Tomlinson, 1993). An alternative hypothesis is that the gaps and bursts arise in cortex from intrinsic circuitry arranged in an opponent fashion.

# 4  CONCLUSION

Common input that causes central peaks on the order of tens of milliseconds wide in spike train CCGs is also responsible for causing the correlation in spike count at the time scale of two second long trials. Long-term correlation due to drifts in responsivity exists but is zero on average across all cell pairs and may represent a source of noise which complicates the accurate measurement of cell-to-cell correlation. The area of the peak of the spike train CCG within a window of $\pm 32$ msec is the basis of a good prediction of the spike count correlation coefficient and provides a less noisy measure of correlation between neurons. Correlated gaps observed in the response to coherent preferred direction motion is consistent with common inhibition and contributes to the area of the spike train CCG peak, and thus to the correlation between spike count. Correlation in spike count is an important factor that can limit the useful pool-size of neuronal ensembles (Zohary et al., 1994; Gawne and Richmond, 1993).

**Acknowledgements**

We thank William T. Newsome, Kenneth H. Britten, Michael N. Shadlen, and J. Anthony Movshon for kindly providing data that was recorded in previous studies and for helpful discussion. This work was funded by the Office of Naval Research and the Air Force Office of Scientific Research. W. B. was supported by the L. A. Hanson Foundation and the Howard Hughes Medical Institute.

**References**

Agmon A, Connors BW (1992) Correlation between intrinsic firing patterns and thalamocortical synaptic responses of neurons in mouse barrel cortex. *J Neurosci* **12**:319–329.

Bair W (1995) *Analysis of Temporal Structure in Spike Trains of Visual Cortical Area MT*. Ph.D. thesis, California Institute of Technology.

Britten KH, Shadlen MN, Newsome WT, Movshon JA (1992) The analysis of visual motion: a comparison of neuronal and psychophysical performance. *J Neurosci* **12**:4745–4765.

Gawne TJ, Richmond BJ (1993) How independent are the messages carried by adjacent inferior temporal cortical neurons? *J Neurosci* **13**:2758–2771.

Ridder WH, Tomlinson A (1993) Suppression of contrasts sensitivity during eyelid blinks. *Vision Res* **33**:1795–1802.

Zohary E, Shadlen MN, Newsome WT (1994) Correlated neuronal discharge rate and its implications for psychophysical performance. *Nature* **370**:140–143.